# Adapting to a Market Shock: Optimal Sequential Market-Making

**Sanmay Das**
Department of Computer Science
Rensselaer Polytechnic Institute
Troy, NY 12180
sanmay@cs.rpi.edu

**Malik Magdon-Ismail**
Department of Computer Science
Rensselaer Polytechnic Institute
Troy, NY 12180
magdon@cs.rpi.edu

## Abstract

We study the profit-maximization problem of a monopolistic market-maker who sets two-sided prices in an asset market. The sequential decision problem is hard to solve because the state space is a function. We demonstrate that the belief state is well approximated by a Gaussian distribution. We prove a key monotonicity property of the Gaussian state update which makes the problem tractable, yielding the first optimal sequential market-making algorithm in an established model. The algorithm leads to a surprising insight: an optimal monopolist can provide *more* liquidity than perfectly competitive market-makers in periods of extreme uncertainty, because a monopolist is willing to absorb initial losses in order to learn a new valuation rapidly so she can extract higher profits later.

## 1 Introduction

Designing markets to achieve certain goals is gaining renewed importance with the prevalence of many novel markets, ranging from prediction markets [13] to markets for e-services [11]. These markets tend to be thin (illiquid) when they first appear. Similarly, when a market shock occurs to the value of an instrument on a financial exchange, thousands of speculative traders suddenly possess new valuations on the basis of which they would like to trade. Periods of uncertainty, like those following a shock, are also periods of illiquidity, so trading may be sparse right after a shock.

This is a chicken-and-egg problem. People do not want to trade in thin markets, and yet, having many people trading is what creates liquidity. These markets therefore need to be bootstrapped into a phase where they are sufficiently liquid to attract trading. This bootstrapping is often achieved through market-makers [12]. Market-makers are responsible for providing liquidity and maintaining order on the exchange. For example, the NYSE designates a single *monopolist* specialist (market-maker) for each stock, while the NASDAQ allows multiple market-makers to compete.

There has been much debate on whether one of these models is better than the other. This debate is again important today for those who are designing new markets. Should they employ a single monopolistic market-maker or multiple competitive market-makers? Alternatively, should the market-maker be based on some other criterion, and if so, what is the optimal design for this agent?

Market makers want to maximize profit, which could run contrary to their "social responsibility" of providing liquidity. A monopolist market maker attempts to maximize expected discounted profits, while competitive (non-colluding) market makers may only expect zero profit, since any profits should be wiped out by competition. Therefore, one would expect markets with competitive market-makers to be of better quality. However, this has not been observed in practice, especially in the well-studied case of the NASDAQ vs. the NYSE [1, 9]. Many explanations have been proposed in the empirical literature, and have explained parts of this phenomenon. One reason that has been speculated about anecdotally but never analyzed formally is the learning aspect of the problem. For

example, the NYSE's promotional literature used to tout the benefits of a monopolist for "maintaining a fair and orderly market" in the face of market shocks [6].

The main challenge to formally analyzing this question is the complexity of the monopolistic market maker's sequential decision problem. The market maker, when setting bid and ask prices, is plagued by a heavily path dependent exploitation-exploration dilema. There is a tradeoff between setting the prices to extract maximum profit from the next trade versus setting the prices to get as much information about the new value of the instrument so as to generate larger profits from future trades. There is no known solution to this sequential decision problem.

We present the first such solution within an established model of market making. We show the surprising fact that a monopolist market maker leads to higher market liquidity in periods of extreme market shock than does a zero-profit competitive market maker. In various single period settings, it has been shown that monopolists can sometimes provide greater liquidity [6] by averaging expected profits across different trade *sizes*. We show for the first time that this can hold true with fixed trade sizes in a *multi-period* setting, because the market-maker is willing to take losses following a shock in order to learn the new valuation more quickly.

## 1.1 Market Microstructure Background

Market microstructure has recently received much attention from a computational perspective [10, 4, 12]. The driving problem of this paper is *price discovery*. Suppose an instrument has just begun trading in a market where different people have different beliefs about its value. An example is shares in the "Barack Obama wins the presidential election" market. These shares should trade at prices that reflect the probability that the event will occur: if the outcome pays off $100, the shares should trade at about $55 if the aggregate public belief is 55% that the event will occur. Similarly, the price of a stock should reflect the aggregate public belief about future cash flows associated with a company. It is well-known that markets are good at aggregating information into prices, but different market structures possess different qualities in this regard. We are concerned with the properties of dealer markets, in which prices are set by one or more market-makers responsible for providing liquidity by taking one side of every trade.

Market-making has been studied extensively in the theoretical market microstructure literature [8, 7, for example], but only recently has the dynamic multi-period problem gained attention [2, 3]. Since we are interested in the problem of how a market-maker learns a value for an asset, we follow the general model of Glosten and Milgrom which abstracts away from the problem of quantities by restricting attention to situations where the market-maker places bid and ask quotes for one unit of the asset at each time step. Das [3] has extended this model to consider the market-maker's learning problem with competitive pricing, while Darley *et al* [2] have used similar modeling for simulations of the NASDAQ. The Glosten and Milgrom model has become a standard model in this area.

Liquidity, which is not easy to quantify, is the prime social concern. In practice, it is a function of the depth of the limit order book. In our models, we measure liquidity using the *bid-ask spread*, or alternatively the probability that a trade will occur. This gives a good indication of the level of informational heterogeneity in the market, and of execution costs. The dynamic behavior of the spread gives insight into the price discovery process.

## 1.2 Our Contribution

We consider the question of optimal *sequential* price-setting in the Glosten-Milgrom model. The market-maker sets bid and ask prices at each trading period[1] and when a trader arrives she has the option of buying or selling at those prices, or of not executing a trade. There are many results relating to the properties of zero-profit (competitive) market-makers [7, 3]. The zero-profit problem is a single-period decision-making problem with online belief updates. Within this same framework, one can formulate the decision problem for a monopolist market-maker who maximizes her total discounted profit as a reinforcement learning problem. The market maker's state is her belief about the instrument value, and her action is to set bid and ask prices. The market maker's actions must trade off profit taking (exploitation) with price discovery (exploration).

The complexity of the sequential problem arises from the complexity of the state space and the fact that the action space is continuous. The state of the market-maker must represent her belief about the true value of the asset being traded. As such, it is a probability density *function*. In a parametric setting, the state space is finite dimensional, but continuous. Even if we assume a Gaussian prior for the market-maker's belief as well as for the beliefs of all the traders, the market-maker's beliefs quickly become a complex product of error functions, and the exact dynamic programming problem becomes intractable.

We solve the Bellman equation for the optimal sequential market maker within the framework of Gaussian state space evolution, a close approximation to the true state space evolution. We present simulation results which testify to how closely the Gaussian framework approximates the true evolution. The Gaussian approximation alone does not alleviate the difficulties associated with reinforcement learning in continuous action and state spaces.[2] However within our setting, we prove a key monotonicity property for the state update. This property allows us to solve for the value function exactly using a single pass dynamic program.

Thus, our first contribution is a complete solution to the optimal sequential market making problem within a Gaussian update framework. Our second contribution relates to the phenomenological implications for market behavior. We obtain the surprising result that in periods of extreme shock, when the market maker has large uncertainty relative to the traders, the monopolist provides greater liquidity than competitive zero-profit market-makers. The monopolist increases liquidity, possibly taking short term losses, in order to learn more quickly, and in doing so offers the better social outcome. Of course, once the monopolist has adapted to the shock, she equilibrates at a higher bid ask spread than the the corresponding zero-profit market maker with the same beliefs.

## 2 The Model and the Sequential Decision Problem

### 2.1 Market Model

At time 0, a shock occurs causing an instrument to attain value $V$ which will be held fixed through time (we consider one instrument in the market). This could represent a real market shock to a stock value (change in public beliefs), an IPO, or the introduction of a new contract in a prediction market. We use a model similar to Das's [3] extension of the Glosten and Milgrom [7] model. We assume that trading is divided into a sequence of discrete trading time steps, each time step corresponding to the arrival of a trader. The value $V$ is drawn from some distribution $g_V(v)$.

The market-maker ($MM$), at each time step $t \geq 0$, sets bid and ask prices $b_t \leq a_t$ at which she is willing to respectively buy and sell one unit. Traders arrive at time-steps $t \geq 0$. Trader $t$ arrives with a noisy estimate $w_t$ of $V$, where $w_t = V + \epsilon_t$. The $\{\epsilon_t\}$ are zero mean i.i.d. random variables with distribution function $F_\epsilon$. We will assume that $F_\epsilon$ is symmetric, so that $F_\epsilon(-x) = 1 - F_\epsilon(x)$. The trader decides whether to trade at either the bid or ask prices depending on the value of $w_t$. The trader will buy at $a_t$ if $w_t > a_t$ (she thinks the instrument is undervalued), sell at $b_t$ if $w_t < b_t$ (she thinks the instrument is overvalued) and do nothing otherwise. $MM$ receives a signal $x_t \in \{+1, 0, -1\}$ indicating whether the trader bought, did nothing or sold. Note that information is conveyed only by the direction of the trade. Information can also be conveyed by the patterns and size of trades, but the present work abstracts away from those considerations.

The market-maker's objective is to maximize profit. In perfect competition, the MM is pushed to setting bid and ask prices that yield zero expected profit. In a monopolistic setting, she wants to optimize the profits she receives over time. As we will see below, this can be a difficult problem to solve. A commonly used alternative is to consider a greedy, or myopically optimal MM who only maximizes her expected profit from the next trade. This is a good approximation for agents with a high discount factor, since they are more concerned with immediate reward. We will consider all three types of market-makers, (1) Zero-profit, (2) Myopic, and (3) Optimal.

## 2.2 State Space

The state space for the MM is determined by MM's belief about the value $V$, described by a density function $p_t$ at time step $t$. The MM decides on actions (bid and ask prices) $(b_t, a_t)$ based on $p_t$. The MM receives signal $x_t \in \{+1, 0, -1\}$ as to whether the trader bought, sold, or did nothing.

Let $q_t(V; b_t, a_t)$ be the probability of receiving signal $x_t$ given bid and ask $(b_t, a_t)$, conditioned on $V$. Assuming that $F_\epsilon$ is continuous at $b_t - V$ and $a_t - V$, a straightforward calculation yields

$$q_t(V; b_t, a_t) = \begin{cases} 1 - F_\epsilon(a_t - V) & x_t = +1, \\ F_\epsilon(a_t - V) - F_\epsilon(b_t - V) & x_t = 0, \\ F_\epsilon(b_t - V) & x_t = -1, \end{cases}$$

or, $q_t(V; b_t, a_t) = F_\epsilon(z_t^+ - V) - F_\epsilon(z_t^- - V)$, where $z_t^+$ and $z_t^-$ are respectively $+\infty, a_t, b_t$ and $a_t, b_t, -\infty$ when $x_t = +1, 0, -1$. The Bayesian update to $p_t$ is then given by $p_{t+1}(v) = p_t(v)\frac{q_t(v; b_t, a_t)}{\mathcal{A}_t}$, where the normalization constant $\mathcal{A}_t = \int_{-\infty}^{\infty} dv\, p_t(v) q_t(v; b_t, a_t)$. Unfolding the recursion gives $p_{t+1}(v) = p_0(v) \prod_{\tau=1}^{t} \frac{q_\tau(v; b_\tau, a_\tau)}{\mathcal{A}_\tau}$

## 2.3 Solving for Market Maker Prices

Let $b_t \leq a_t$, and let $r_t$ be the expected profit at time $t$. The expected discounted return is then $R = \sum_{t=0}^{\infty} \gamma^t r_t$ where $0 < \gamma < 1$ is the discount factor. The optimal MM maximizes $R$. We can compute $r_t$ as $r_t = \int_{-\infty}^{\infty} dv\, vF_\epsilon(-v)\,(p_t(v + b_t) + p_t(a_t - v))$. $r_t$ decomposes into two terms which can be identified as the bid and ask side profits, $r_t = r_t^{bid}(b_t) + r_t^{ask}(a_t)$. In perfect competition, $MM$ should not be expecting any profit on either the bid or ask side. This is because if the contrary were true, a competing MM could place bid or ask prices so as to obtain less profit, wiping out $MM$'s advantage. This should hold at every time step. Hence the $MM$ will set bid and ask prices such that $r_t^{bid}(b_t) = 0$ and $r_t^{ask}(a_t) = 0$. Solving for $b_t, a_t$, we find that $b_t$ and $a_t$ must satisfy the following fixed point equations (these are also derived for the case of Gaussian noise by Das [3]),

$$b_t = \frac{\int_{-\infty}^{\infty} dv\, vp_t(v)F_\epsilon(b_t - v)}{\int_{-\infty}^{\infty} dv\, p_t(v)F_\epsilon(b_t - v)} = E_{p_t}[V|x_t = -1], a_t = \frac{\int_{-\infty}^{\infty} dv\, vp_t(v)F_\epsilon(v - a_t)}{\int_{-\infty}^{\infty} dv\, p_t(v)F_\epsilon(v - a_t)} = E_{p_t}[V|x_t = +1]$$

(assuming the denominators, which are the conditional probabilities of hitting the bid or ask are non-zero). The myopic monopolist maximizes $r_t$. For the typical case of well behaved distributions $p_t(v)$ and $F_\epsilon$, the bid and ask returns display a single maximum. In this case, we can obtain $b_t^{myp}$ and $a_t^{myp}$ by setting the derivatives to zero (we assume the functions are well behaved so that the derivatives are defined). Letting $f_\epsilon(x) = F_\epsilon'(x)$ be the density function for the noise $\epsilon_t$, $b_t^{myp}$ and $a_t^{myp}$ satisfy the fixed point equations

$$b_t = \frac{\int_{-\infty}^{\infty} dv\, p_t(v)(vf_\epsilon(b_t - v) - F_\epsilon(b_t - v))}{\int_{-\infty}^{\infty} dv\, p_t(v)f_\epsilon(b_t - v)}, a_t = \frac{\int_{-\infty}^{\infty} dv\, p_t(v)(vf_\epsilon(a_t - v) + F_\epsilon(v - a_t))}{\int_{-\infty}^{\infty} dv\, p_t(v)f_\epsilon(a_t - v)}$$

The optimal strategy for MM is not as easy to obtain. When $\gamma$ is large, the expected discounted return $R$ could be significantly higher than the myopic return. The optimal MM might choose to sacrifice short term return for a substantially larger return over the long term. The only reason to do this is if choosing a sub-optimal short term strategy will lead to a significant decrease in the uncertainty in $V$ (which translates to a narrowing of the probability distribution $p_t(v)$). MM can then exploit this more certain information regarding $V$ in the longer term.

The optimal strategy for the MM is encapsulated in the Bellman equation for the value functional (where the state $p_t$, is a *function*, $(b_t, a_t)$ is the action, and $\pi$ is a policy):

$$V(p_t; \pi) = E[r_0|p_t, b_t^\pi(p_t), a_t^\pi(p_t)] + \gamma E[V(p_{t+1}; \pi)|p_t, b_t^\pi(p_t), a_t^\pi(p_t)]$$

This equation reflects the fact that the MM's expected profit is a function of both her immediate expected return, and her future state, which is also affected by her bid and ask prices. The fact that $V$ is a value *functional* leads to numerous technical problems when solving this Bellman equation. The problem is heavily path dependent with the number of paths being exponential in the number of trading periods. To make this tractable, we use a Gaussian approximation for the state space evolution.

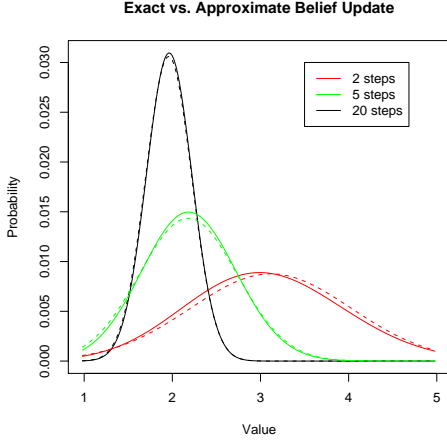

$$\begin{aligned}
I(\alpha, \beta) &= \int_{-\infty}^{\infty} dx\, N(x) \int_{-\infty}^{\alpha - \beta x} dy\, N(y) \\
&= \Phi\left(\frac{\alpha}{\sqrt{1+\beta^2}}\right), \\
J(\alpha, \beta) &= \int_{-\infty}^{\infty} dx\, x \cdot N(x) \int_{-\infty}^{\alpha - \beta x} dy\, N(y) \\
&= -\sqrt{\frac{\beta^2}{1+\beta^2}} \cdot N\left(\frac{\alpha}{\sqrt{1+\beta^2}}\right), \\
K(\alpha, \beta) &= \int_{-\infty}^{\infty} dx\, x^2 \frac{e^{-x^2/2}}{\sqrt{2\pi}} \int_{-\infty}^{\alpha - \beta x} dy\, \frac{e^{-y^2/2}}{\sqrt{2\pi}} \\
&= I(\alpha, \beta) - \frac{\alpha\beta^2}{(1+\beta^2)^{3/2}} \cdot N\left(\frac{\alpha}{\sqrt{1+\beta^2}}\right) \\
L(\alpha, \beta) &= I(\alpha, \beta) - K(\alpha, \beta) \\
A(z^+, z^-) &= I\left(\frac{z^+ - \mu_t}{\sigma_\epsilon}, \rho_t\right) - I\left(\frac{z^- - \mu_t}{\sigma_\epsilon}, \rho_t\right) \\
B(z^+, z^-) &= J\left(\frac{z^+ - \mu_t}{\sigma_\epsilon}, \rho_t\right) - J\left(\frac{z^- - \mu_t}{\sigma_\epsilon}, \rho_t\right) \\
C(z^+, z^-) &= L\left(\frac{z^+ - \mu_t}{\sigma_\epsilon}, \rho_t\right) - L\left(\frac{z^- - \mu_t}{\sigma_\epsilon}, \rho_t\right)
\end{aligned}$$

Figure 1: Gaussian state update (dashed) versus true state update (solid) illustrating that the Gaussian approximation is valid.

Figure 2: Gaussian integrals and normalization constants used in the derivation of the DP and the state updates.

## 2.4 The Gaussian Approximation

From a Gaussian prior and performing Bayesian updates, one expects that the state distribution will be closely approximated by a Gaussian (see Figure 1). Thus, forcing the MM to maintain a Gaussian belief over the true value at each time $t$ should give a good approximation to the true state space evolution, and the resulting optimal actions should closely match the true optimal actions. In making this reduction, we reduce the state space to a two parameter function class parameterized by the mean and variance, $(\mu_t, \sigma_t^2)$. The value function is independent of $\mu_t$ (hence dependent only on $\sigma_t$), and the optimal action is of the form $b_t = \mu_t - \delta_t, a_t = \mu_t + \delta_t$. Thus,

$$V(\sigma_t) = \max_\delta \left\{ r_t(\sigma_t, \delta) + \gamma E[V(\sigma_{t+1})|\delta] \right\} \tag{1}$$

To compute the expectation on the RHS, we need the probabilistic dynamics in the (approximate) Gaussian state space, i.e., we need the evolution of $\mu_t, \sigma_t$.

Let $N(\cdot), \Phi(\cdot)$ denote the standard normal density and distribution. Let $p_t(v) = \frac{1}{\sigma_t} N\left(\frac{v - \mu_t}{\sigma_t}\right)$ be Gaussian with mean $\mu_t$ and variance $\sigma_t^2$. Assume that the noise is also Gaussian with variance $\sigma_\epsilon^2$, so $F_\epsilon(x) = \Phi(\frac{x}{\sigma_\epsilon})$. At time $t+1$, after the Bayesian update, we have

$$p_{t+1} = \frac{1}{A} \cdot \frac{1}{\sigma_t} N\left(\frac{v - \mu_t}{\sigma_t}\right) \left[ \Phi\left(\frac{z^+ - v}{\sigma_\epsilon}\right) - \Phi\left(\frac{z^- - v}{\sigma_\epsilon}\right) \right].$$

The normalization constant $A(z^+, z^-)$ is given in Figure 2, and $z_t^+$ and $z_t^-$ are respectively $+\infty, a_t, b_t$ and $a_t, b_t, -\infty$ when $x_t = +1, 0, -1$. The updates $\mu_{t+1}$ and $\sigma_{t+1}^2$ are obtained from $E_{p_{t+1}}[V] = \int dv\, v p_{t+1}(v)$ and $E_{p_{t+1}}[V^2] = \int dv\, v^2 p_{t+1}(v)$. After some tedious algebra (see supplementary information), we obtain

$$\mu_{t+1} = \mu_t + \sigma_t \cdot \frac{B}{A}, \tag{2}$$

$$\sigma_{t+1}^2 = \sigma_t^2 \left( 1 - \frac{AC + B^2}{A^2} \right). \tag{3}$$

Figure 2 gives the expressions for $A, B, C$.

**Theorem 2.1** (Monotonic state update)**.** $\sigma_{t+1}^2 \leq \sigma_t^2$ *(see supplementary information for proof).*

Establishing that $\sigma_t$ is decreasing in $t$ allows us to solve the dynamic program efficiently (note that the property of decreasing variance is well-known for the case of an update to a Gaussian prior when the observation is also Gaussian – we are showing this for threshold observations).

## 2.5 Solving the Bellman Equation

We now return to the Bellman equation (1). In light of Theorem 2.1, the RHS of this equation is dependent only on states $\sigma_{t+1}$ that are strictly smaller than the state $\sigma_t$ on the LHS. We can thus solve this problem numerically by computing $V(0)$ and then building up the solution for a fine grid on the real line. We use linear interpolation between previously computed points if the variance update leads to a point not on the grid.

We need to explicitly construct the states on the RHS with respect to which the expectation is being taken. The expectation is with respect to the future state $\sigma_{t+1}$, which depends directly on the trade outcome $x_t \in \{-1, 0, +1\}$. We define $\rho_t = \sigma_t/\sigma_\epsilon$ and $q = \delta_t/\sigma_\epsilon\sqrt{1 + \rho_t^2}$, where $a_t = \mu_t + \delta_t$ and $b_t = \mu_t - \delta_t$. The following table sumarizes some of the useful quantities:

| $x_t$ | Prob. | $\mu_{t+1}$ | $\sigma_{t+1}$ |
|-------|-------|-------------|----------------|
| +1 | $1 - \Phi(q_t)$ | $\mu_t + \kappa_t\sigma_t$ | $\alpha_t\sigma_t$ |
| 0 | $2\Phi(q_t) - 1$ | $\mu_t$ | $\beta_t\sigma_t$ |
| −1 | $1 - \Phi(q_t)$ | $\mu_t - \kappa_t\sigma_t$ | $\alpha_t\sigma_t$ |

where

$$\alpha_t^2 = 1 - \frac{\rho_t^2 N(q_t)(N(q_t) - q_t[1 - \Phi(q_t)])}{(1 + \rho_t^2)(1 - \Phi(q_t))^2}$$

$$\beta_t^2 = 1 - \frac{2\rho_t^2 q_t N(q_t)}{(1 + \rho_t^2)(2\Phi(q_t) - 1)}$$

$$\kappa_t = \sqrt{\frac{\rho_t^2}{1 + \rho_t^2}\frac{N(q_t)}{1 - \Phi(q_t)}}$$

Note that $q_t > 0$, $\alpha_t, \beta_t < 1$ and $\kappa_t > 0$. We can now compute $E[V(\sigma_{t+1}|\delta_t)]$ as

$$2(1 - \Phi(q_t))V(\alpha_t\sigma_t) + (2\Phi(q_t) - 1)V(\beta_t\sigma_t).$$

This allows us to complete the specification for the Bellman equation (with $x = \rho_t^2$ where $\rho_t = \frac{\sigma}{\sigma_\epsilon}$ is the MM's information disadvantage)

$$V(x; \sigma_\epsilon) = \max_q \left\{ 2\sigma_\epsilon^2\sqrt{1 + x}\left(q(1 - \Phi(q)) - \frac{x}{1 + x}N(q)\right) \right.$$

$$\left. + \gamma\left[2(1 - \Phi(q))V(\alpha^2(x, q)x; \sigma_\epsilon) + (2\Phi(q) - 1)V(\beta^2(x, q)x; \sigma_\epsilon)\right] \right\}$$

where $\alpha^2(x, q)$ and $\beta^2(x, q)$ are as defined above with $\rho_t^2 = x$ and $q_t = q$.

We define the optimal action $q^*(x)$ as the value of $q$ that maximizes the $RHS$. When $x = 0$, the myopic and optimal $MM$ coincide, and so we have that $V(0) = \frac{2q^*(1 - \Phi(q^*))}{1 - \gamma}$, where $q^* = q^*(0) \approx 0.7518$ satisfies $q^*N(q^*) = 1 - \Phi(q^*)$. Note that if we only maximize the first term in the value function, we obtain the myopic action $q^{myp}(\rho)$, satisfying the fixed point equation: $q^{myp} = (1 + \rho_t^2)\frac{1 - \Phi(q^{myp})}{N(q^{myp})}$. There is a similarly elegant solution for the zero-profit MM under the Gaussian assumption, obtained by setting $r_t = 0$, yielding the fixed point equation: $q^{zero} = \frac{\rho_t^2}{1 + \rho_t^2}\frac{N(q^{zero})}{1 - \Phi(q^{zero})}$. 10 standard fixed point iterations are sufficient to solve these equations accurately.

## 3 Experimental Results

First, we validate the Gaussian approximation by simulating a market as follows. The initial value $V$ is drawn from a Gaussian with mean 0 and standard deviation $\sigma$, and we set the discount rate $\gamma = 0.9$. Each simulation consists of 100 trading periods at which point discounted returns become negligible. At each trading step $t$, a new trader arrives with a valuation $w_t \sim N(V, 1)$ (Gaussian with mean $V$ and variance 1). We report results averaged over more than 10,000 simulations, each with a randomly sampled value of $V$.

In each simulation, the market-maker's state updates are given by the Gaussian approximation (2), (3), according to which she sets bid and ask prices. The trader at time-step $t$ trades by comparing $w_t$ to $b_t, a_t$. We simulate the outcomes of the optimal, myopic, and zero-profit MMs. An alternative

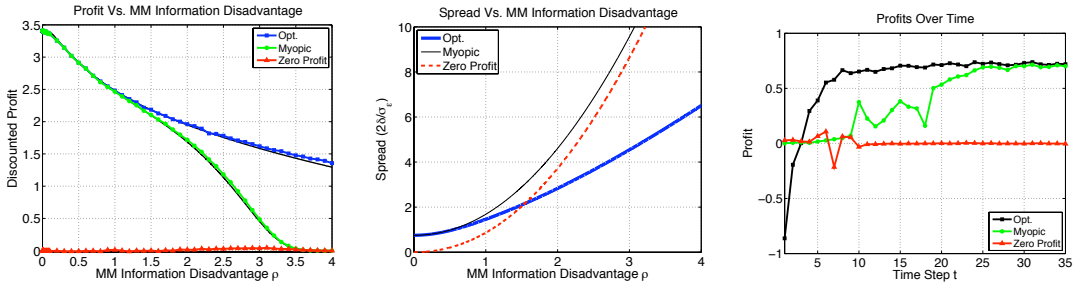

(a) Realized vs theoretical value function in the Gaussian approximation (thin black line). The realized closely matches the theoretical, validating the Gaussian framework.

(b) Bid-ask spreads as a function of the MM information disadvantage $\rho$ indicating that once $\rho$ exceeds about 1.5, the monopolist offers the greatest liquidity.

(c) Realized average return as a function of time: the monopolist is willing to take significant short term loss to improve future profits as a result of better price discovery.

Figure 3: MM Properties derived from the solution of the Bellman equation.

is to maintain the exact state as a product of error functions, and extract the mean and variance for computing the optimal action. This is computationally prohibitive, and leads to no significant differences. If the real world conformed to the MM's belief, a new value $V_t$ would be drawn from $N(\mu_t, \sigma_t)$ at each trading period $t$, and then the trader would receive a sample $w_t \sim N(V_t, 1)$. All our computations are exact within this "Gaussian" world, however the point here is to test the degree to which the Gaussian and real worlds differ.

The ideal test of our optimal MM is against the true optimal for the real world, which is intractable. However, if we find that the theoretical value function for the optimal MM in the Gaussian world matches the realized value function in the real world, then we have strong, though not necessarily conclusive, evidence for two conclusions: (1) The Gaussian world is a good approximation to the real world, otherwise the realized and theoretical value functions would not coincide; (2) Since the two worlds are nearly the same, the optimal MM in the Gaussian world should closely match the true optimal. Figure 3(a) presents results which show that the realized and theoretical value functions are essentially the same, presenting the desired evidence (note that with independent updates, the posterior should be asymptotically Gaussian). Figure 3(a) also demonstrates that the optimal significantly outperforms the myopic market-maker. Figure 3(b) shows how the bid-ask spread will behave as a function of the MM information disadvantage.

Some phenomenological properties of the market are shown in Figure 4.[3] For a starting MM information disadvantage of $\rho = 3$, the optimal MM initially has significantly lower spread, even compared with the zero profit market-maker. The reason for this outcome is illustrated in Figure 3(c) where we see that the optimal market maker is offering lower spreads and taking on significant initial loss to be compensated later by significant profits due to better price discovery. At equilibrium the optimal MM's spread and the myopic spread are equal, as expected.

## 4 Discussion

Our solution to the Bellman equation for the optimal monopolistic MM leads to the striking conclusion that the optimal MM is willing to take early losses by offering *lower* spreads in order to make significantly higher profits later (Figures 3(b,c) and 4). This is quantitative evidence that the optimal MM offers more liquidity than a zero-profit MM after a market shock, especially when the MM is at a large information disadvantage. In this regime, exploration is more important than exploitation. Competition may actually impede the price discovery process, since the market makers would have no incentive to take early losses for better price discovery – competitive pricing is not necessarily *informationally* efficient (there are quicker ways for the market to "learn" a new valuation).

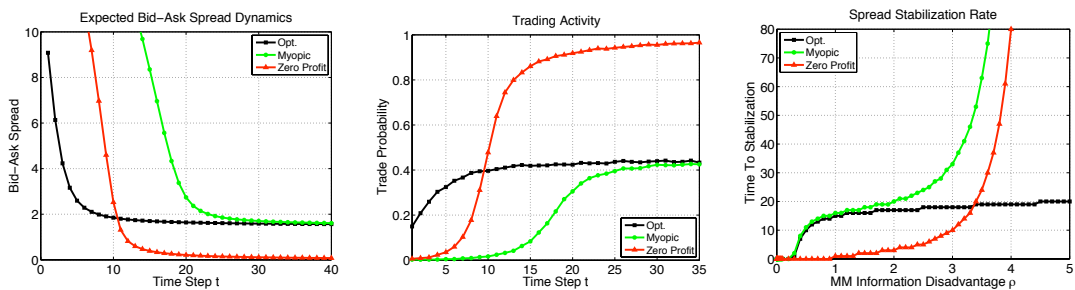

(a) Realized spread over time ($\sigma = 3$). The optimal MM starts with lowest spread, and converges quickest to equilibrium.

(b) Liquidity over time ($\sigma = 3$), measured by probability of a trade. Initial liquidity is higher for the optimal MM.

(c) Time to spread stabilization. When MM's information disadvantage increases, the optimal MM is significantly better.

Figure 4: Realized market properties based on simulating the three MMs.

Our solution is based on reducing a functional state space to a finite-dimensional one in which the Bellman equation can be solved efficiently. When the state is a probability distribution, updated according to independent events, we expect the Gaussian approximation to closely match the real state evolution. Hence, our methods may be generally applicable to problems of this form.

While this paper presents a stylized model, simple trading models have been shown to produce rich market behavior in many cases (for example, [5]). The results presented here are an example of the kinds of insights that can be be gained from studying market properties in these models while approaching agent decision problems from the perspective of machine learning. At the same time, this paper is not purely theoretical. The eventual algorithm we present is easy to implement, and we are in the process of evaluating this algorithm in test prediction markets. Another direction we are pursuing is to endow the traders with intelligence, so they may learn the true value too. We believe the Gaussian approximation admits a solution for a monopolistic market-maker and adaptive traders.

## Footnotes

[1]The MM is willing to buy at the bid price and sell at the ask price.

[2]Where one has to resort to unbounded value iteration methods whose convergence and uniqueness properties are little understood.

[3]With both zero-profit and optimal MMs we reproduce one of the key findings of Das [3]: the market exhibits a two-regime behavior. Price jumps are immediately followed by a regime of high spreads (the price-discovery regime), and then when the market-maker learns the new valuation, the market settles into an equilibrium regime of lower spreads (the efficient market regime).

## References

[1] W.G. Christie and P.H. Schulz. Why do NASDAQ market makers avoid odd-eighth quotes? *J. Fin.*, 49(5), 1994.

[2] V. Darley, A. Outkin, T. Plate, and F. Gao. Sixteenths or pennies? Observations from a simulation of the NASDAQ stock market. In *IEEE/IAFE/INFORMS Conf. on Comp. Intel. for Fin. Engr.*, 2000.

[3] S. Das. A learning market-maker in the Glosten-Milgrom model. *Quant. Fin.*, 5(2):169–180, April 2005.

[4] E. Even-Dar, S.M. Kakade, M. Kearns, and Y. Mansour. (In)stability properties of limit order dynamics. In *Proc. ACM Conf. on Elect. Comm.*, 2006.

[5] J.D. Farmer, P. Patelli, and I.I Zovko. The predictive power of zero intelligence in financial markets. *PNAS*, 102(11):2254–2259, 2005.

[6] L.R. Glosten. Insider trading, liquidity, and the role of the monopolist specialist. *J. Bus.*, 62(2), 1989.

[7] L.R. Glosten and P.R. Milgrom. Bid, ask and transaction prices in a specialist market with heterogeneously informed traders. *J. Fin. Econ.*, 14:71–100, 1985.

[8] S.J. Grossman and M.H. Miller. Liquidity and market structure. *J. Fin.*, 43:617–633, 1988.

[9] Roger D. Huang and Hans R. Stoll. Dealer versus auction markets: A paired comparison of execution costs on NASDAQ and the NYSE. *J. Fin. Econ.*, 41(3):313–357, 1996.

[10] S.M. Kakade, M. Kearns, Y. Mansour, and L. Ortiz. Competitive algorithms for VWAP and limit-order trading. In *Proc. ACM Conf. on Elect. Comm.*, pages 189–198, 2004.

[11] Juong-Sik Lee and Boleslaw Szymanski. Auctions as a dynamic pricing mechanism for e-services. In Cheng Hsu, editor, *Service Enterprise Integration*, pages 131–156. Kluwer, New York, 2006.

[12] D. Pennock and R. Sami. Computational aspects of prediction markets. In N. Nisan, T. Roughgarden, E. Tardos, and V.V. Vazirani, editors, *Algorithmic Game Theory*. Cambridge University Press, 2007.

[13] Justin Wolfers and Eric Zitzewitz. Prediction markets. *J. Econ. Persp.*, 18(2):107–126, 2004.

